# Catastrophic interference in connectionist networks: Can it be predicted, can it be prevented?

**Robert M. French**
Computer Science Department
Willamette University
Salem, Oregon 97301
french@willamette.edu

## 1  OVERVIEW

Catastrophic forgetting occurs when connectionist networks learn new information, and by so doing, forget all previously learned information. This workshop focused primarily on the causes of catastrophic interference, the techniques that have been developed to reduce it, the effect of these techniques on the networks' ability to generalize, and the degree to which prediction of catastrophic forgetting is possible. The speakers were Robert French, Phil Hetherington (Psychology Department, McGill University, het@blaise.psych.mcgill.ca), and Stephan Lewandowsky (Psychology Department, University of Oklahoma, lewan@constellation.ecn.uoknor.edu).

## 2  PROTOTYPE BIASING AND FORCED SEPARATION OF HIDDEN-LAYER REPRESENTATIONS

French indicated that catastrophic forgetting is at its worst when high representation overlap at the hidden layer combines with significant teacher-output error. He showed that techniques to reduce this overlap tended to decrease catastrophic forgetting. Activation sharpening, a technique that produces representations having a few highly active nodes and many low-activation nodes, was shown to be effective because it reduced representation overlap. However, this technique was ineffective for large data sets because creating localized representations reduced the number of possible hidden-layer representations. Hidden layer representations that were more distributed but still highly separated were needed. French introduced prototype biasing, a technique that uses a separate network to learn a prototype for each teacher pattern. Hidden-layer representations of new items are made to resemble their prototypes. Each representation is also "separated" from the representation of the previously encountered pattern according to the difference between the respective teachers. This technique produced hidden-layer representations that were both distributed and well separated. The result was a significant decrease in catastrophic forgetting.

# 3   ELIMINATING CATASTROPHIC INTERFERENCE BY PRETRAINING

Hetherington presented a technique that consisted of prior training of the network on a large body of items of the same type as the new items in the sequential learning task. Hetherington measured the degree of actual forgetting, as did all of the authors, by the method of savings, i.e., by determining how long the network takes to relearn the original data set that has been "erased" by learning the new data. He showed that when networks are given the benefit of relevant prior knowledge, the representations of the new items are constrained *naturally* and interference may be virtually eliminated. The previously encoded knowledge causes new items to be encoded in more orthogonal manner (i.e., with less mutual overlap) than in a naive (i.e., non-pretrained) network. The resulting decrease in representation overlap produced the virtual elimination of catastrophic forgetting.

Hetherington also presented another technique that substantially reduced catastrophic interference in the sequential learning task. Learning of new items takes place in a windowed, or overlapping fashion. In other words, as new items are learned the network continues learning on the most recently presented items.

# 4   THE RELATIONSHIP BETWEEN INTERFERENCE AND GENERALIZATION

Lewandowsky examined the hypothesis that generalization is compromised in networks that had been "manipulated" to decrease catastrophic interference by creating semi-distributed (i.e., only partially overlapping) representations at the hidden layer. He gave a theoretical analysis of the relationship between interference and generalization and then presented results from several different simulations using semi-distributed representations. His conclusions were that semi-distributed representations can significantly reduce catastrophic interference in backpropagation networks without diminishing their generalization abilities. This was only true, however, for techniques (e.g., activation sharpening) that reduced interference by creating a more robust final weight pattern but that did not change the activation surfaces of the hidden units. On the other hand, in models where interference is reduced by eliminating overlap between receptive fields of static hidden units (i.e., by altering their response surface), generalization abilities are impaired.

In addition, Lewandowsky presented a technique that relied on orthogonalizing the input vectors to a standard backpropagation network by converting standard asymmetric input vectors (each node at 0 or 1) to symmetric input vectors (each input node at -1 or 1). This technique was also found to significantly reduce catastrophic interference.